# Large Margin Multi-channel Analog-to-Digital Conversion with Applications to Neural Prosthesis

**Amit Gore and Shantanu Chakrabartty**
Department of Electrical and Computer Engineering
Michigan State University
East Lansing, MI 48823
{goreamit,shantanu}@egr.msu.edu

## Abstract

A key challenge in designing analog-to-digital converters for cortically implanted prosthesis is to sense and process high-dimensional neural signals recorded by the micro-electrode arrays. In this paper, we describe a novel architecture for analog-to-digital (A/D) conversion that combines $\Sigma\Delta$ conversion with spatial de-correlation within a single module. The architecture called multiple-input multiple-output (MIMO) $\Sigma\Delta$ is based on a min-max gradient descent optimization of a regularized linear cost function that naturally lends to an A/D formulation. Using an online formulation, the architecture can adapt to slow variations in cross-channel correlations, observed due to relative motion of the micro-electrodes with respect to the signal sources. Experimental results with real recorded multi-channel neural data demonstrate the effectiveness of the proposed algorithm in alleviating cross-channel redundancy across electrodes and performing data-compression directly at the A/D converter.

## 1 Introduction

Design of cortically implanted neural prosthetic sensors (CINPS)is an active area of research in the rapidly emerging field of brain machine interfaces (BMI) [1, 2]. The core technology used in these sensors are micro-electrode arrays (MEAs) that facilitate real-time recording from thousands of neurons simultaneously. These recordings are then actively processed at the sensor (shown in Figure 1) and transmitted to an off-scalp neural processor which controls the movement of a prosthetic limb [1]. A key challenge in designing implanted integrated circuits (IC) for CINPS is to efficiently process high-dimensional signals generated at the interface of micro-electrode arrays [3, 4]. Sensor arrays consisting of more than 1000 recording elements are common [5, 6] which significantly increase the transmission rate at the sensor. A simple strategy of recording, parallel data conversion and transmitting the recorded neural signals ( at a sampling rate of 10 KHz) can easily exceed the power dissipation limit of $80mW/cm^2$ determined by local heating of biological tissue [7]. In addition to increased power dissipation, high-transmission rate also adversely affects the real-time control of neural prosthesis [3].

One of the solutions that have been proposed by several researchers is to perform compression of the neural signals directly at the sensor, to reduce its wireless transmission rate and hence its power dissipation [8, 4]. In this paper we present an approach where de-correlation or redundancy elimination is performed directly at analog-to-digital converter. It has been shown that neural cross-talk and common-mode effects introduces unwanted redundancy at the output of the electrode array [4]. As a result, neural signals typically occupy only a small sub-space within the high-dimensional space spanned by the micro-electrode signals. An optimal strategy for designing a multi-channel analog-to-digital converter is to identify and operate within the sub-space spanned by the neural signals and in the process eliminate cross-channel redundancy. To achieve this goal, in this paper we pro-

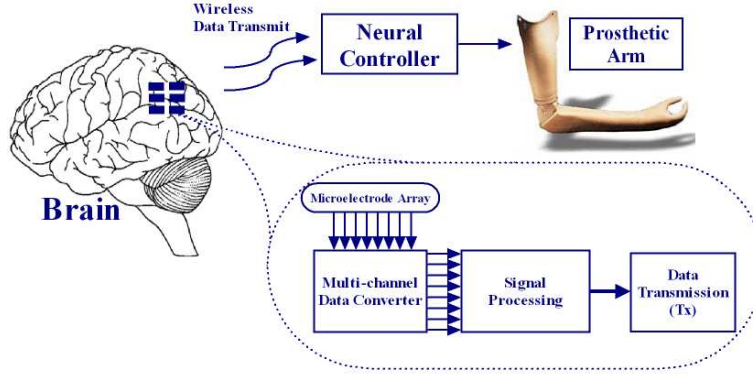

Figure 1: Functional architecture of a cortically implanted neural prosthesis illustrating the interface of the data converter to micro-electrode arrays and signal processing modules

pose to use large margin principles [10], which have been highly successful in high-dimensional information processing [11, 10]. Our approach will be to formalize a cost function consisting of $L_1$ norm of the internal state vector whose gradient updates naturally lends to a digital time-series expansion. Within this framework the correlation distance between the channels will be minimized which amounts to searching for signal spaces that are maximally separated from each other.

The architecture called multiple-input multiple-output (MIMO) $\Sigma\Delta$ converter is the first reported data conversion technique to embed large margin principles. The approach, however, is generic and can be extended to designing higher order ADC. To illustrate the concept of MIMO A/D conversion, the paper is organized as follows: section 2 introduces a regularization framework for the proposed MIMO data converter and introduces the min-max gradient descent approach. Section 3 applies the technique to simulated and recorded neural data. Section 4 concludes with final remarks and future directions.

## 2 Regularization Framework and Generalized $\Sigma\Delta$ Converters

In this section we introduce an optimization framework for deriving MIMO $\Sigma\Delta$ converters. For the sake of simplicity we will first assume that the input to converter is a $M$ dimensional vector $\mathbf{x} \in \mathcal{R}^M$ where each dimension represents a single channel in the multi-electrode array. It is also assumed that the vector $\mathbf{x}$ is stationary with respect to discrete time instances $n$. The validity and limitation of this assumption is explained briefly at the end of this section. Also denote a linear transformation matrix $\mathbf{A} \in \mathcal{R}^{M \times M}$ and an regression weight vector $\mathbf{w} \in \mathcal{R}^M$. Consider the following optimization problem

$$\min_{\mathbf{w}} f(\mathbf{w}, \mathbf{A}) \tag{1}$$

where

$$f(\mathbf{w}, \mathbf{A}) = |\mathbf{w}|^T \mathbf{1} - \mathbf{w}^T \mathbf{A} \mathbf{x} \tag{2}$$

and $\mathbf{1}$ represents a column vector whose elements are unity. The cost function in equation 2 consists of two factors: the first factor is an $L_1$ regularizer which constrains the norm of the vector $\mathbf{w}$ and the second factor that maximizes the correlation between vector $\mathbf{w}$ and an input vector $\mathbf{x}$ transformed using a linear projection denoted by matrix $\mathbf{A}$. The choice of $L_1$ norm and the form of cost function in equation (2) will become clear when we present its corresponding gradient update rule. To ensure that the optimization problem in equation 1 is well defined, the norm of the input vector $||\mathbf{x}||_\infty \leq 1$ will be assumed to be bounded.

Under bounded condition, the closed form solution to optimization problem in equation 1 can be found to be $\mathbf{w}^* = \mathbf{0}$. From the perspective of A/D conversion we will show that the iterative steps leading towards solution to the optimization problem in equation 1 are more important than the final solution itself. Given an initial estimate of the state vector $\mathbf{w}[0]$ the online gradient descent step for

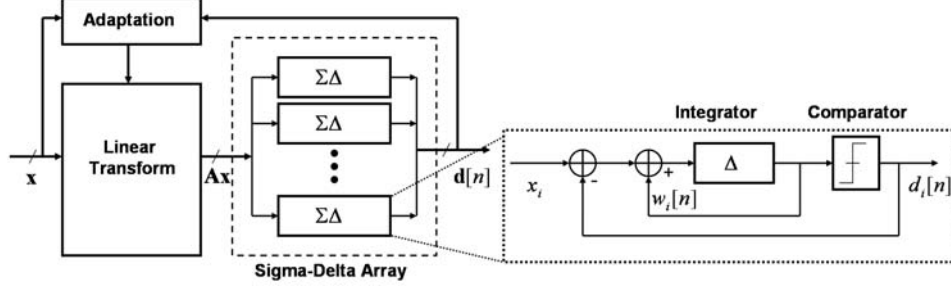

Figure 2: Architecture of the proposed first-order MIMO $\Sigma\Delta$ converter.

minimizing 1 at iteration $n$ is given by

$$\mathbf{w}[n] = \mathbf{w}[n-1] - \eta \frac{\partial f}{\partial \mathbf{w}} \tag{3}$$

where $\eta > 0$ is defined as the learning rate. The choice of $L_1$ norm in optimization function in equation 1 ensures that for $\eta > 0$ the iteration 3 exhibits oscillatory behavior around the solution $\mathbf{w}^*$. Combining equation (3) with equation (2) the following recursion is obtained:

$$\mathbf{w}[n] = \mathbf{w}[n-1] + \eta(\mathbf{A}\mathbf{x} - \mathbf{d}[n]) \tag{4}$$

where

$$\mathbf{d}[n] = \text{sgn}(\mathbf{w}[n-1]) \tag{5}$$

and $\text{sgn}(u)$ denotes an element-wise signum operation such that $\mathbf{d}[n] \in \{+1, -1\}^M$ represents a digital time-series. The iterations in 3 represents the recursion step for $M$ first-order $\Sigma\Delta$ converters [9] coupled together by the linear transform $\mathbf{A}$. If we assume that the norm of matrix $||\mathbf{A}||_\infty \leq 1$ is bounded, it can be shown that $||\mathbf{w}_\infty|| < 1 + \eta$. Following N update steps the recursion given by equation 4 yields

$$\mathbf{A}\mathbf{x} - \frac{1}{N}\sum_{n=1}^{N}\mathbf{d}[n] = \frac{1}{\eta N}(\mathbf{w}[N] - \mathbf{w}[0]) \tag{6}$$

which using the bounded property of $\mathbf{w}$ asymptotically leads to

$$\frac{1}{N}\sum_{n=1}^{N}\mathbf{d}[n] \longrightarrow \mathbf{A}\mathbf{x} \tag{7}$$

as $N \to \infty$.

Therefore consistent with the theory of $\Sigma\Delta$ conversion [9] the moving average of vector digital sequence $\mathbf{d}[n]$ converges to the transformed input vector $\mathbf{A}\mathbf{x}$ as the number of update steps $N$ increases. It can also be shown that $N$ update steps yields a digital representation which is $log_2(N)$ bits accurate.

## 2.1  Online adaptation and compression

The next step is to determine the form of the matrix $\mathbf{A}$ which parameterize the family of linear transformations spanning the signal space. The aim of optimizing for $\mathbf{A}$ is to find multi-channel signal configuration that is maximally separated from each other. For this purposes we denote one channel as a reference relative to which all distances/correlations will be measured. This is unlike independent component analysis (ICA) based approaches [12], where the objective is to search for maximally independent signal space including the reference channel. Even though several forms of the matrix $\mathbf{A} = [a_{ij}]$ can be chosen, for reasons which will discussed later in this paper the matrix $\mathbf{A}$ is chosen to be a lower triangular matrix such that $a_{ij} = 0; i < j$ and $a_{ij} = 1; i = j$. The choice of a lower triangular matrix ensures that the matrix $\mathbf{A}$ is always invertible. It also implies

that the first channel is unaffected by the proposed transform $\mathbf{A}$ and will be the reference channel. The problem of compression or redundancy elimination is therefore to optimize the cross-elements $a_{ij}, i \neq j$ such that the cross-correlation terms in optimization function given by equation 1 are minimized. This can be written as a min-max optimization criterion where an inner optimization performs analog-to-digital conversion, where as the outer loop adapts the linear transform matrix $\mathbf{A}$ such as to maximize the margin of separation between the respective signal spaces. This can be denoted by the following equation:

$$\max_{a_{ij} i \neq j} \left( \min_{\mathbf{w}} f(\mathbf{w}, \mathbf{A}) \right) \tag{8}$$

In conjunction with the gradient descent steps in equation 4 the update rule for elements of $\mathbf{A}$ follows a gradient ascent step given by

$$a_{ij}[n] = a_{ij}[n-1] - \varepsilon u_i[n] x_j; \forall i > j \tag{9}$$

where $\varepsilon$ is a learning rate parameter. The update rule in equation 9 can be made amenable to hardware implementation by considering only the sign of the regression vector $\mathbf{w}[n]$ and the input vector $\mathbf{x}$ as

$$a_{ij}[n] = a_{ij}[n-1] - \varepsilon d_i[n] \operatorname{sign}(x_j); \forall i > j. \tag{10}$$

The update rule in equation 10 bears strong resemblance to online update rules used in independent component analysis (ICA) [12, 13]. The difference with the proposed technique however is the integrated data conversion coupled with spatial decorrelation/compression. The output of the MIMO $\Sigma\Delta$ converter is a digital stream whose pulse density is proportional to the transformed input data vector as

$$\frac{1}{N} \sum_{n=1}^{N} \mathbf{d}[n] \longrightarrow \mathbf{A}[n]\mathbf{x} \tag{11}$$

By construction the MIMO converter produces a digital stream whose pulse-density contains only non-redundant information. To achieve compression some of the digital channels can be discarded (based on their relative energy criterion ) and can also be shut down to conserve power. The original signal can be reconstructed from the compressed digital stream by applying an inverse transformation $\mathbf{A}^{-1}$ as

$$\widehat{x} = \frac{1}{N} \mathbf{A}[n]^{-1} \left( \sum_{n=1}^{N} \mathbf{d}[n] \right). \tag{12}$$

An advantage of using a lower triangular form for the linear transformation matrix $\mathbf{A}$ with its diagonal elements as unity, is that its inverse always well-defined. Thus signal reconstruction using the output of the analog-to-digital converter is also always well defined. Since the transformation matrix $\mathbf{A}$ is continually being updated, the information related to the linear transform also needs to be periodically transmitted to ensure faithful reconstruction at the external prosthetic controller. However, analogous to many naturally occurring signal the underlying statistics of multi-dimensional signal changes slowly as the signal itself. Therefore the transmission of the matrix $\mathbf{A}$ needs to be performed at a relatively slower rate than the transmission of the compressed neural signals.

Similar to conventional $\Sigma\Delta$ conversion [9], the framework for MIMO $\Sigma\Delta$ can be extended to time-varying input vector under the assumption of high oversampling criterion [9]. For a MIMO A/D converter oversampling ratio (OSR) is defined by the ratio of the update frequency $f_s$ and the maximum Nyquist rate amongst all elements of the input vector $\mathbf{x}[n]$. The resolution of the MIMO $\Sigma\Delta$ is also determined by the OSR as $\log_2(OSR)$ and during the oversampling period the input signal vector can be assumed to be approximately stationary. For time-varying input vector

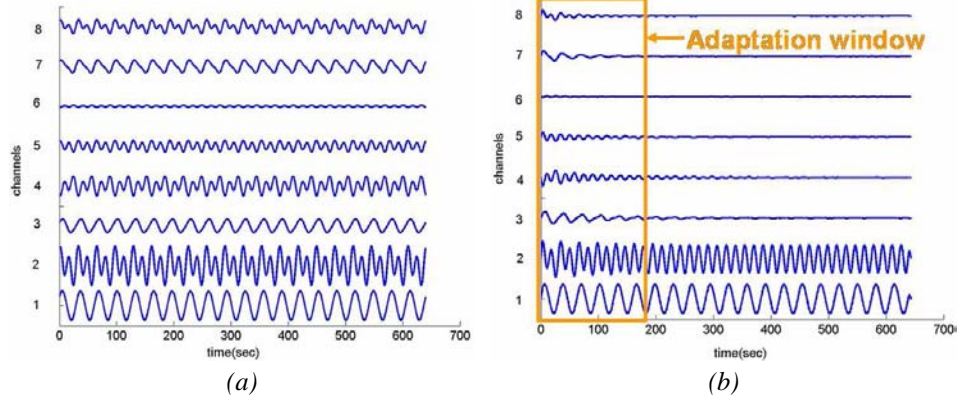

*(a)*                 *(b)*

Figure 3: Functional verification of MIMO $\Sigma\Delta$ converter on artificially generated multi-channel data (a) Data presented to the MIMO $\Sigma\Delta$ converter (b) Analog representation of digital output produced by MIMO converter

$\mathbf{x}[n] = \{x_j[n]\}, j = 1, .., M$ the matrix update equation in equation 10 can be generalized after $N$ steps as

$$\frac{1}{N}a_{ij}[N] = \varepsilon\frac{1}{N}\sum_{n=1}^{N} d_i[n]sgn(x_j[n]); \forall i > j.\tag{13}$$

Thus if the norm of the matrix $\mathbf{A}$ is bounded, then asymptotically $N \to \infty$ the equation 13 imply that the cross-channel correlation between the digital output and the sign of the input signal approaches zero. This is similar to formulations in ICA where higher-order de-correlation is achieved using non-linear functions of random variables [12].

The architecture for the MIMO $\Sigma\Delta$ converter illustrating recursions (4) and (11) is shown in Figure 2. As shown in the Figure 2 the regression vectors $\mathbf{w}[n]$ within the framework of MIMO $\Sigma\Delta$ represents the output of the $\Sigma\Delta$ integrator. All the adaptation and linear transformation steps can be implemented using analog VLSI with adaptation steps implemented either using multiplying digital-to-analog converters or floating gates synapses. Even though any channel can be chosen as a reference channel, our experiments indicate that the channel with maximum cross-correlation and maximum signal power serves as the best choice.

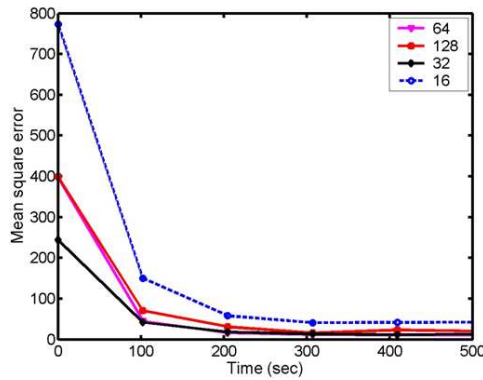

Figure 4: Reconstruction performance in terms of mean square error computed using artificial data for different OSR

# 3 Results

The functionality of the proposed MIMO sigma-delta converter was verified using artificially generated data and with real multi-channel recorded neural data. The first set of experiments simulated an artificially generated 8 channel data. Figure 3(a) illustrates the multi-channel data where each channel was obtained by random linear mixing of two sinusoids with frequency 20Hz and 40Hz. The multi-channel data was presented to a MIMO sigma delta converter implemented in software. The equivalent analog representation of the pulse density encoded digital stream was obtained using a moving window averaging technique with window size equal to the oversampling ratio (OSR). The resultant analog representation of the ADC output is shown in 3(b). It can be seen in the figure that after initial adaptation steps the output corresponding to first two channels converges to the fundamental sinusoids, where as the rest of the digital streams converged to an equivalent zero output. This simple experiment demonstrates the functionality of MIMO sigma-delta in eliminating cross-channel redundancy. The first two digital streams were used to reconstruct the original recording using equation 12. Figure 4 shows the reconstruction error averaged over a time window of 2048 samples showing that the error indeed converges to zero, as the MIMO converter adapts. The Figure 4 also shows the error curves for different OSR. It can be seen that even though better reconstruction error can be achieved by using higher OSR, the adaptation procedure compensates for errors introduced due to low resolution. In fact the reconstruction performance is optimal for intermediate OSR.

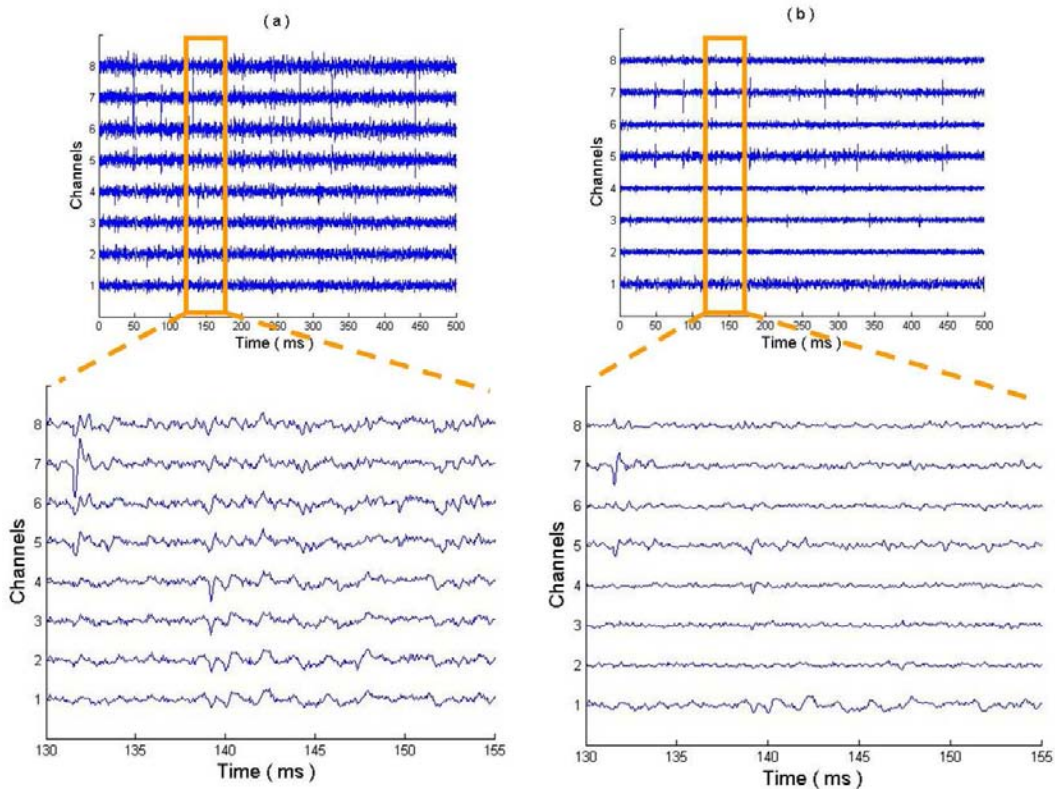

Figure 5: Functional verification of the MIMO sigma-delta converter for multi-channel neural data: (a) Original multichannel data (b) analog representation of digital output produced by the converter

The multi-channel experiments were repeated with an eight channel neural data recorded from dorsal cochlear nucleus of adult guinea pigs. The data was recorded at a sampling rate of 20KHz and at a resolution of 16 bits. Figure 5(a) shows a clip of multi-channel recording for duration of 0.5 seconds. It can be seen from highlighted portion of Figure 5(a) that the data exhibits high degree of cross-channel correlation. Similar to the first set of experiments the MIMO converter eliminates spatial redundancy between channels as shown by the analog representation of the reconstructed output

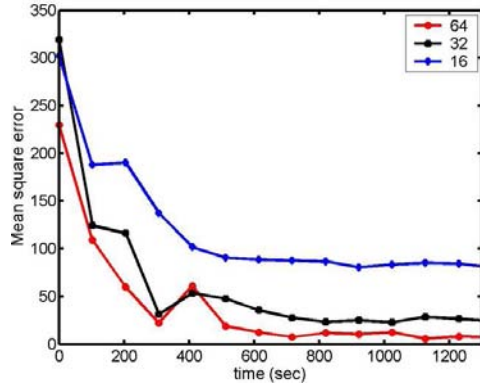

Figure 6: Reconstruction performance in terms of mean square error computed using neural data for different OSR

in Figure 5(b). An interesting observation in this experiment is that even though the statistics of the input signals varies in time as shown in Figure 5 (a) and (b), the transformation matrix **A** remains relatively stationary during the duration of the conversion, which is illustrated through the reconstruction error graph in Figure 6. This validates the principle of operation of the MIMO conversion where the multi-channel neural recording lie on a low-dimensional manifold whose parameters are relatively stationary with respect to the signal statistics.

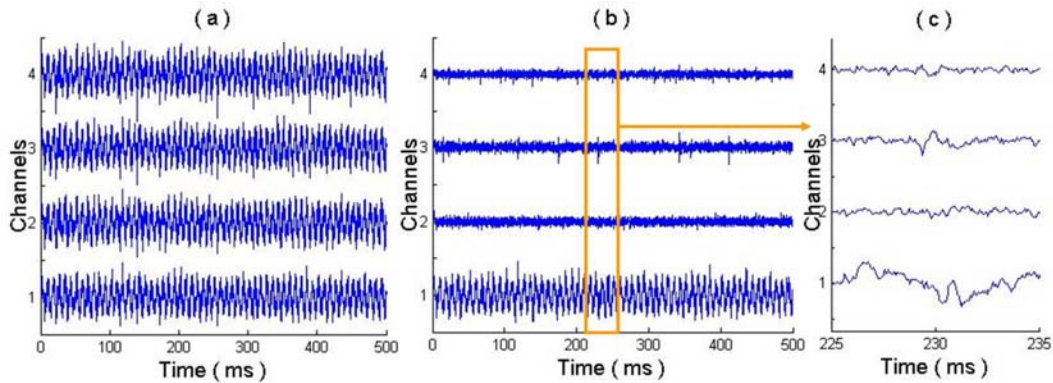

Figure 7: Demonstration of common-mode rejection performed by MIMO $\Sigma\Delta$: (a) Original multi-channel signal at the input of converter (b) analog representation of the converter output (c) a magnified clip of the output produced by the converter illustrating preservation of neural information.

The last set of experiments demonstrate the ability of the proposed MIMO converter to reject common mode disturbance across all the channels. Rejection of common-mode signal is one of the most important requirement for processing neural signals whose amplitude range from $50\mu V$ - $500\mu V$, where as the common-mode interference resulting from EMG or electrical coupling could be as high as $10mV$ [14]. Therefore most of the micro-electrode arrays use bio-potential amplifiers for enhancing signal-to-noise ratio and common-mode rejection. For this set of experiments, the recorded neural data obtained from the previous experiment was contaminated by an additive 60Hz sinusoidal interference of amplitude $1mV$. The results are shown in Figure 7 illustrating that the reference channel absorbs all the common-mode disturbance where as the neural information is preserved in other channels. In fact theoretically it can be shown that the common-mode rejection ratio for the proposed MIMO ADC is dependent only on the OSR and is given by $20\log_{10} OSR$.

# 4 Conclusion

In this paper we presented a novel MIMO analog-to-digital conversion algorithm with application to multi-channel neural prosthesis. The roots of the algorithm lie within the framework of large margin principles, where the data converter maximizes the relative distance between signal space corresponding to different channels. Experimental results with real multi-channel neural data demonstrate the effectiveness of the proposed method in eliminating cross-channel redundancy and hence reducing data throughput and power dissipation requirements of a multi-channel biotelemetry sensor. There are several open questions that needs to be addressed as a continuation of this research which includes extension of the algorithm second-order $\Sigma\Delta$ architectures, embedding of kernels into the ADC formulation and reformulation of the update rule to perform ICA directly on the ADC.

### Acknowledgments

This work is supported by grant from National Institute of Health (R21NS047516-01A2). The authors would also like to thank Prof. Karim Oweiss for providing multi-channel neural data for the MIMO ADC experiments.

# References

[1] Kennedy, P. R., R. A. Bakay, M. M. Moore, K. Adams, and J. Goldwaithe. Direct control of a computer from the human central nervous system. IEEE Trans Rehabil Eng 8:198-202, 2000.

[2] J. Carmena, M. Lebedev, R. E. Crist, J. E. ODoherty, D. M. Santucci, D. Dimitrov, P. Patil, C. S. Henriquez, and M. A. Nicolelis, Learning to control a brain-machine interface for reaching and grasping by primates, PLoS Biol., vol. 1, no. 2, pp. 193208, Nov. 2003.

[3] G. Santhanam, S. I. Ryu, B. M. Yu, and K. V. Shenoy, High information transmission rates in a neural prosthetic system, in Soc. Neurosci., 2004, Program 263.2.

[4] K. Oweiss, D. Anderson, M. Papaefthymiou, Optimizing Signal Coding in Neural Interface System-on-a-Chip Modules, IEEE Conf. on EMBS, pp. 2016-2019, Sept. 2003.

[5] K. Wise et. al., Wireless Implantable Microsystems: High-Density Electronic Interfaces to the Nervous System, Proc. of the IEEE, Vol.: 92-1, pp: 7697, Jan. 2004.

[6] Maynard EM, Nordhausen CT, Normann RA, The Utah intracortical electrode array: a recording structure for potential brain computer interfaces. Electroencephalogr Clin Neurophysiol 102: 228239, 1997.

[7] T. M. Seese, H. Harasaki, G. M. Saidel, and C. R. Davies, Characterization of tissue morphology, angiogenesis, and temperature in adaptive response of muscle tissue to chronic heating, Lab Investigation, vol. 78, no. 12, pp. 15531562, Dec. 1998.

[8] R. R. Harrison, A low-power integrated cicuit for adaptive detection of action potentials in noisy signals, in Proc. 25th Ann. Conf. IEEE EMBS, Cancun, Mexico, Sep. 2003, pp. 33253328.

[9] J. C. Candy and G. C. Temes, Oversampled methods for A/D and D/A conversion, in Oversampled Delta-Sigma Data Converters. Piscataway, NJ: IEEE Press, 1992, pp. 1- 29.

[10] Vapnik, V. The Nature of Statistical Learning Theory, New York: Springer-Verlag, 1995.

[11] Girosi, F., Jones, M. and Poggio, T. Regularization Theory and Neural Networks Architectures, Neural Computation, vol. 7, pp 219-269, 1996.

[12] Hyvrinen, A. Survey on independent component, analysis. Neural Computing Surveys, 2:94128, 1999.

[13] A. Celik, M. Stanacevic and G. Cauwenberghs, Gradient Flow Independent Component Analysis in Micropower VLSI, Adv. Neural Information Processing Systems (NIPS'2005), Cambridge: MIT Press, 18, 2006

[14] Pedram Mohseni and Khalil Najafi. A fully integrated neural recording amplifier with DC input stabilization. Biomedical Engineering, IEEE Transactions on Volume 51, Issue 5, May 2004.
